# Bayesian Self-Organization

**Alan L. Yuille**
Division of Applied Sciences
Harvard University
Cambridge, MA 02138

**Stelios M. Smirnakis**
Lyman Laboratory of Physics
Harvard University
Cambridge, MA 02138

**Lei Xu** *
Dept. of Computer Science
HSH ENG BLDG, Room 1006
The Chinese University of Hong Kong
Shatin, NT
Hong Kong

## Abstract

Recent work by Becker and Hinton (Becker and Hinton, 1992) shows a promising mechanism, based on maximizing mutual information assuming spatial coherence, by which a system can self-organize itself to learn visual abilities such as binocular stereo. We introduce a more general criterion, based on Bayesian probability theory, and thereby demonstrate a connection to Bayesian theories of visual perception and to other organization principles for early vision (Atick and Redlich, 1990). Methods for implementation using variants of stochastic learning are described and, for the special case of linear filtering, we derive an analytic expression for the output.

## 1  Introduction

The input intensity patterns received by the human visual system are typically complicated functions of the object surfaces and light sources in the world. It

seems probable, however, that humans perceive the world in terms of surfaces and objects (Nakayama and Shimojo, 1992). Thus the visual system must be able to extract information from the input intensities that is relatively independent of the actual intensity values. Such abilities may not be present at birth and hence must be learned. It seems, for example, that binocular stereo develops at about the age of two to three months (Held, 1987).

Becker and Hinton (Becker and Hinton, 1992) describe an interesting mechanism for self-organizing a system to achieve this. The basic idea is to assume spatial coherence of the structure to be extracted and to train a neural network by maximizing the mutual information between neurons with disjoint receptive fields. For binocular stereo, for example, the surface being viewed is assumed flat (see (Becker and Hinton, 1992) for generalizations of this assumption) and hence has spatially constant disparity. The intensity patterns, however, do not have any simple spatial behaviour. Adjusting the synaptic strengths of the network to maximize the mutual information between neurons with non-overlapping receptive fields, for an ensemble of images, causes the neurons to extract features that are spatially coherent - thereby obtaining the disparity [fig.1].

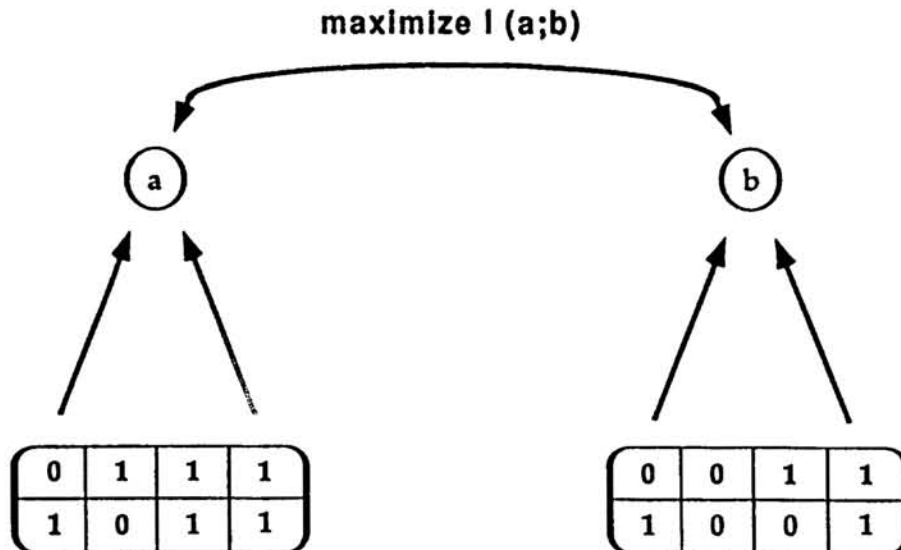

Figure 1: In Hinton and Becker's initial scheme (Becker and Hinton, 1992), maximization of mutual information between neurons with spatially disjoint receptive fields leads to disparity tuning, provided they train on spatially coherent patterns (i.e. those for which disparity changes slowly with spatial position)

Workers in computer vision face a similar problem of estimating the properties of objects in the world from intensity images. It is commonly stated that vision is ill-posed (Poggio et al, 1985) and that prior assumptions about the world are needed to obtain a unique perception. It is convenient to formulate such assumptions by the use of Bayes' theorem $P(S|D) = P(D|S)P(S)/P(D)$. This relates the proba-

bility $P(S|D)$ of the scene $S$ given the data $D$ to the prior probability of the scene $P(S)$ and the imaging model $P(D|S)$ ($P(D)$ can be interpreted as a normalization constant). Thus a vision theorist (see (Clark and Yuille, 1990), for example) determines an imaging model $P(D|S)$, picks a set of plausible prior assumptions about the world $P(S)$ (such as *natural constraints* (Marr, 1982)), applies Bayes' theorem, and then picks an interpretation $S^*$ from some statistical estimator of $P(S|D)$ (for example, the maximum a posteriori (MAP) estimator $S^* = ARG\{MAX_S P(S|D)\}$.)

An advantage of the Bayesian approach is that, by nature of its probabilistic formulation, it can be readily related to learning with a teacher (Kersten et al, 1987). It is unclear, however, whether such a teacher will always be available. Moreover, from Becker and Hinton's work on self-organization, it seems that a teacher is not always necessary. This paper proposes a way for generalizing the self-organization approach, by starting from a Bayesian perspective, and thereby relating it to Bayesian theories of vision. The key idea is to force the activity distribution of the outputs to be close to a pre-specified prior distribution $P_p(S)$. We argue that this approach is in the same spirit as (Becker and Hinton, 1992), because we can choose the prior distribution to enforce spatial coherence, but it is also more general since many other choices of the prior are possible. It also has some relation to the work performed by Atick and Redlich (Atick and Redlich, 1990) for modelling the early visual system.

We will take the viewpoint that the prior $P_p(S)$ is assumed known in advance by the visual system (perhaps by being specified genetically) and will act as a self-organizing principle. Later we will discuss ways that this might be relaxed.

## 2    Theory

We assume that the input $D$ is a function of a *signal* $\Sigma$ that the system wants to determine and a *distractor* $N$ [fig.2]. For example $\Sigma$ might correspond to the disparities of a pair of binocular stereo images and $N$ to the intensity patterns. The distribution of the inputs is $P_D(D)$ and the system *assumes* that the signal $\Sigma$ has distribution $P_p(\Sigma)$.

Let the output of the system be $S = G(D, \gamma)$ where $G$ is a function of a set of parameters $\gamma$ to be determined. For example, the function $G(D, \gamma)$ could be represented by a multi-layer perceptron with the $\gamma$'s being the synaptic weights. By approximation theory, it can be shown that a large variety of neural networks can approximate any input-output function arbitrarily well given enough hidden nodes (Hornik et al, 1989).

The aim of self-organizing the network is to ensure that the parameters $\gamma$ are chosen so that the outputs $S$ are as close to the $\Sigma$ as possible. We claim that this can be achieved by adjusting the parameters $\gamma$ so as to make the derived distribution of the outputs $P_{DD}(S : \gamma) = \int \delta(S - G(D, \gamma)) P_D(D)[dD]$ as close as possible to $P_p(S)$.

This can be seen to be a consistency condition for a Bayesian theory as from Bayes formula we obtain the equation:

$$\int P(S|D) P_D(D)[dD] = \int P(D|S) P_p(S)[dD] = P_p(S). \qquad (1)$$

which is equivalent to our condition, provided we choose to identify $P(S|D)$ with $\delta(S - G(D, \gamma))$.

To make this more precise we must define a measure of similarity between the two distributions $P_p(S)$ and $P_{DD}(S : \gamma)$. An attractive measure is the Kullback-Leibler distance (the entropy of $P_{DD}$ relative to $P_p$):

$$KL(\gamma) = \int P_{DD}(S : \gamma) \log \frac{P_{DD}(S : \gamma)}{P_p(S)} [dS]. \tag{2}$$

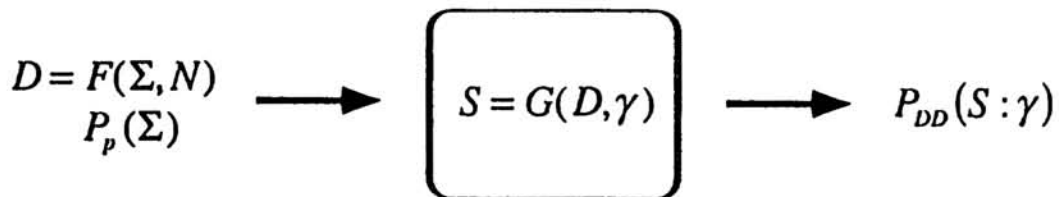

$$KL(\gamma) = \int P_{DD}(S : \gamma) \log \left( \frac{P_{DD}(S : \gamma)}{P_p(S)} \right) dS$$

Figure 2: The parameters $\gamma$ are adjusted to minimize the Kullback-Leibler distance between the prior ($P_p$) distribution of the true signal ($\Sigma$) and the derived distribution ($P_{DD}$) of the network output ($S$).

This measure can be divided into two parts: (i) $- \int P_{DD}(S : \gamma) \log P_p(S)[dS]$ and (ii) $\int P_{DD}(S : \gamma) \log P_{DD}(S : \gamma)[dS]$. The second term encourages variability of the output while the first term forces similarity to the prior distribution.

Suppose that $P_p(S)$ can be expressed as a Markov random field (i.e. the spatial distribution of $P_p(S)$ has a local neighbourhood structure, as is commonly assumed in Bayesian models of vision). Then, by the Hammersely-Clifford theorem, we can write $P_p(S) = e^{-\beta E_p(S)}/Z$ where $E_p(S)$ is an energy function with local connections (for example, $E_p(S) = \sum_i (S_i - S_{i+1})^2$), $\beta$ is an inverse temperature and $Z$ is a normalization constant.

Then the first term can be written (Yuille et al, 1992) as

$$- \int P_{DD}(S : \gamma) \log P_p(S)[dS] = \beta \langle E_p(G(D, \gamma)) \rangle_D + \log Z. \tag{3}$$

We can ignore the $\log Z$ term since it is a constant (independent of $\gamma$). Minimizing the first term with respect to $\gamma$ will therefore try to minimize the energy of the outputs averaged over the inputs - $\langle E_p(G(D, \gamma)) \rangle_D$ - which is highly desirable (since it has a close connection to the minimal energy principles in (Poggio et al, 1985, Clark and Yuille, 1990)). It is also important, however, to avoid the trivial solution $G(D, \gamma) = 0$ as well as solutions for which $G(D, \gamma)$ is very small for most inputs. Fortunately these solutions are discouraged by the second term : $\int P_{DD}(D, \gamma) \log P_{DD}(D, \gamma)[dD]$, which corresponds to the negative entropy of the derived distribution of the network output. Thus, its minimization with respect to $\gamma$ is a maximum entropy principle which will encourage variability in the outputs $G(D, \gamma)$ and hence prevent the trivial solutions.

## 3    Reformulating for Implementation.

Our theory requires us to minimize the Kullback-Leibler distance, equation 2, with respect to $\gamma$. We now describe two ways in which this could be implemented using variants of stochastic learning. First observe that by substituting the form of the derived distribution into equation 2 and integrating out the $S$ variable we obtain:

$$KL(\gamma) = \int P_D(D) \log \frac{P_{DD}(G(D, \gamma) : \gamma)}{P_p(G(D, \gamma))}[dD].$$ (4)

Assuming a representative sample $\{D^\mu : \mu \,\epsilon\, \Lambda\}$ of inputs we can approximate $KL(\gamma)$ by $\sum_{\mu \epsilon \Lambda} \log[P_{DD}(G(D^\mu, \gamma) : \gamma)/P_p(G(D^\mu, \gamma))]$. We can now, in principle, perform stochastic learning using backpropagation: pick inputs $D^\mu$ at random and update the weights $\gamma$ using $\log[P_{DD}(G(D^\mu, \gamma) : \gamma)/P_p(G(D^\mu, \gamma))]$ as the error function.

To do this, however, we need expressions for $P_{DD}(G(D^\mu, \gamma) : \gamma)$ and its derivative with repect to $\gamma$. If the function $G(D, \gamma)$ can be restricted to being 1-1 (increasing the dimensionality of the output space if necessary) then we can obtain (Yuille et al, 1992) analytic expressions $P_{DD}(G(D, \gamma) : \gamma) = P_D(D)/|\det(\partial G/\partial D)|$ and $(\partial \log P_{DD}(G(D, \gamma) : \gamma)/\partial \gamma) = -(\partial G/\partial D)^{-1}(\partial^2 G/\partial D \partial \gamma)$, where $[-1]$ denotes the matrix inverse. Alternatively we can perform additional sampling to estimate $P_{DD}(G(D, \gamma) : \gamma)$ and $(\partial \log P_{DD}(G(D, \gamma) : \gamma)/\partial \gamma)$ directly from their integral representations. (This second approach is similar to (Becker and Hinton, 1992) though they are only concerned with estimating the first and second moments of these distributions.)

## 4    Connection to Becker and Hinton.

The Becker and Hinton method (Becker and Hinton, 1992) involves maximizing the mutual information between the output of two neuronal units $S_1, S_2$ [fig.1]. This is given by :

$$I(S_1, S_2) = - < \log P_{DD}(S_1) > - < \log P_{DD}(S_2) > + < \log P_{DD}(S_1, S_2) >$$

where the first two terms correspond to maximizing the entropies of $S_1$ and $S_2$ while the last term forces $S_1 \approx S_2$.

By contrast, our version tries to minimize the quantity :

$$< \log P_{DD}(S_1, S_2) > - \int \log P_p(S_1, S_2) P_{DD}(S_1, S_2)[d\vec{S}].$$

If we then ensure that $P_p(S_1, S_2) = \delta(S_1 - S_2)$ our second term will force $S_1 \approx S_2$ and our first term will maximize the entropy of the joint distribution of $S_1, S_2$. We argue that this is effectively the same as (Becker and Hinton, 1992) since maximizing the joint entropy of $S_1, S_2$ with $S_1$ constrained to equal $S_2$ is equivalent to maximizing the individual entropies of $S_1$ and $S_2$ with the same constraint.

To be more concrete, we consider Becker and Hinton's implementation of the mutual information maximization principle in the case of units with continuous outputs. They assume that the outputs of units $1, 2$ are Gaussian [1] and perform steepest descent to maximize the symmetrized form of the mutual information between $S_1$ and $S_2$:

$$I_{S_1;S_2} = \log \frac{V(S_1)}{V(S_1 - S_2)} + \log \frac{V(S_2)}{V(S_1 - S_2)} = \log V(S_1) + \log V(S_2) - 2 \log V(S_1 - S_2) \tag{5}$$

where $V()$ stands for variance over the set of inputs. They assume that the difference between the two outputs can be expressed as uncorrelated additive noise, $S_1 = S_2 + N$. We reformalize their criterion as maximizing $E_{BH}(V(S_2), V(N))$ where

$$E_{BH}(V(S_2), V(N)) = \log\{V(S_2) + V(N)\} + \log V(S_2) - 2 \log V(N). \tag{6}$$

For our scheme we make similar assumptions about the distributions of $S_1$ and $S_2$. We see that $< \log P_{DD}(S_1, S_2) >= - \log\{< S_1^2 >< S_2^2 > - < S_1 S_2 >^2\} = - \log\{V(S_2)V(N)\}$ (since $< S_1 S_2 >=< (S_2 + N)S_2 >= V(S_2)$ and $< S_1^2 >= V(S_2) + V(N)$). Using the prior distribution $P_p(S_1, S_2) \approx e^{-\tau(S_1 - S_2)^2}$ our criterion corresponds to minimizing $E_{YSX}(V(S_2), V(N))$ where:

$$E_{YSX}(V(S_2), V(N)) = - \log V(S_2) - \log V(N) + \tau V(N). \tag{7}$$

It is easy to see that maximizing $E_{BH}(V(S_2), V(N))$ will try to make $V(S_2)$ as large as possible and force $V(N)$ to zero (recall that, by definition, $V(N) \geq 0$). Minimizing our energy will try to make $V(S_2)$ as large as possible and will force $V(N)$ to $1/\tau$ (recall that $\tau$ appears as the inverse of the variance of a Gaussian prior distribution for $S_1 - S_2$ so making $\tau$ large will force the prior distribution to approach $\delta(S_1 - S_2)$.) Thus, provided $\tau$ is very large, our method will have the same effect as Becker and Hinton's.

## 5 Application to Linear Filtering.

We now describe an analysis of these ideas for the case of linear filtering. Our approach will be contrasted with the traditional Wiener filter approach.

Consider a process of the form $D(\vec{x}) = \Sigma(\vec{x}) + N(\vec{x})$ where $D(\vec{x})$ denotes the input to the system, $\Sigma(\vec{x})$ is the true signal which we would like to predict, and $N(\vec{x})$ is the noise corrupting the signal. The resulting Wiener filter $A_w(\vec{x})$ has fourier transform $\hat{A}_w = \Phi_{\Sigma,\Sigma}/(\Phi_{\Sigma,\Sigma} + \Phi_{N,N})$ where $\Phi_{\Sigma,\Sigma}$ and $\Phi_{N,N}$ are the power spectrum of the signal and the noise respectively.

By contrast, let us extract a linear filter $A_b$ by applying our criterion. In the case that the noise and signal are independent zero mean Gaussian distributions this filter can be calculated explicitly (Yuille et al, 1992). It has fourier transform with squared magnitude given by $|\hat{A}_b|^2 = \Phi_{\Sigma,\Sigma}/(\Phi_{\Sigma,\Sigma} + \Phi_{N,N})$. Thus our filter can be thought of as the square root of the Wiener filter.

It is important to realize that although our derivation assumed additive Gaussian noise our system would not need to make any assumptions about the noise distribution. Instead our system would merely need to assume that the filter was linear and then would *automatically* obtain the "correct" result for the additive Gaussian noise case. We conjecture that the system might detect non-Gauusian noise by finding it impossible to get zero Kullback-Liebler distance with the linear ansatz.

## 6   Conclusion

The goal of this paper was to introduce a Bayesian approach to self-organization using prior assumptions about the signal as an organizing principle. We argued that it was a natural generalization of the criterion of maximizing mutual information assuming spatial coherence (Becker and Hinton, 1992). Using our principle it should be possible to self-organize Bayesian theories of vision, assuming that the priors are known, the network is capable of representing the appropriate functions and the learning algorithm converges. There will also be problems if the probability distributions of the true signal and the distractor are too similar.

If the prior is not correct then it may be possible to detect this by evaluating the goodness of the Kullback-Leibler fit after learning [2]. This suggests a strategy whereby the system increases the complexity of the priors until the Kullback-Leibler fit is sufficiently good (this is somewhat similar to an idea proposed by Mumford (Mumford, 1992)). This is related to the idea of competitive priors in vision (Clark and Yuille, 1990). One way to implement this would be for the prior probability itself to have a set of adjustable parameters that would enable it to adapt to different classes of scenes. We are currently (Yuille et al, 1992) investigating this idea and exploring its relationships to Hidden Markov Models.

Ways to implement the theory, using variants of stochastic learning, were described. We sketched the relation to Becker and Hinton.

As an illustration of our approach we derived the filter that our criterion would give for filtering out additive Gaussian noise (possibly the only analytically tractable case). This had a very interesting relation to the standard Wiener filter.

## Acknowledgements

We would like to thank DARPA for an Air Force contract F49620-92-J-0466. Conversations with Dan Kersten and David Mumford were highly appreciated.

## Footnotes

*Lei Xu was a research scholar in the Division of Applied Sciences at Harvard University while this work was performed.

[1]We assume for simplicity that these Gaussians have zero mean.

[2]This is reminiscent of Barlow's suspicious coincidence detectors (Barlow, 1993), where we might hope to determine if two variables $x$ & $y$ are independent or not by calculating the Kullback-Leibler distance between the joint distribution $P(x,y)$ and the product of the individual distributions $P(x)P(y)$.

## References

J.J. Atick and A.N. Redlich. "Towards a Theory of Early Visual Processing". *Neural Computation*. Vol. 2, No. 3, pp 308-320. Fall. 1990.

H.B. Barlow. "What is the Computational Goal of the Neocortex?" To appear in: **Large scale neuronal theories of the brain**. Ed. C. Koch. MIT Press. 1993.

S. Becker and G.E. Hinton. "Self-organizing neural network that discovers surfaces in random-dot stereograms". *Nature*, Vol 355. pp 161-163. Jan. 1992.

J.J. Clark and A.L. Yuille. **Data Fusion for Sensory Information Processing Systems**. Kluwer Academic Press. Boston/Dordrecht/London. 1990.

R. Held. "Visual development in infants". In **The encyclopedia of neuroscience**, vol. 2. Boston: Birkhauser. 1987.

K. Hornik, S. Stinchocombe and H. White. "Multilayer feed-forward networks are universal approximators". *Neural Networks* 4, pp 251-257. 1991.

D. Kersten, A.J. O'Toole, M.E. Sereno, D.C. Knill and J.A. Anderson. "Associative learning of scene parameters from images". *Optical Society of America*, Vol. 26, No. 23, pp 4999-5006. 1 December, 1987.

D. Marr. **Vision**. W.H. Freeman and Company. San Francisco. 1982.

D. Mumford. "Pattern Theory: a unifying perspective". Dept. Mathematics Preprint. Harvard University. 1992.

K. Nakayama and S. Shimojo. "Experiencing and Perceiving Visual Surfaces". *Science*. Vol. 257, pp 1357-1363. 4 September. 1992.

T. Poggio, V. Torre and C. Koch. "Computational vision and regularization theory". *Nature*, 317, pp 314-319. 1985.

A.L. Yuille, S.M. Smirnakis and L. Xu. "Bayesian Self-Organization". Harvard Robotics Laboratory Technical Report. 1992.
